# The Role of Top-down and Bottom-up Processes in Guiding Eye Movements during Visual Search

**Gregory J. Zelinsky**[†‡]**, Wei Zhang**[‡]**, Bing Yu**[‡]**, Xin Chen**[†*]**, Dimitris Samaras**[‡]

Dept. of Psychology[†], Dept. of Computer Science[‡]
State University of New York at Stony Brook
Stony Brook, NY 11794
Gregory.Zelinsky@stonybrook.edu[†], xichen@ic.sunysb.edu[*]
{wzhang,ybing,samaras}@cs.sunysb.edu[‡]

## Abstract

To investigate how top-down (TD) and bottom-up (BU) information is weighted in the guidance of human search behavior, we manipulated the proportions of BU and TD components in a saliency-based model. The model is biologically plausible and implements an artificial retina and a neuronal population code. The BU component is based on feature-contrast. The TD component is defined by a feature-template match to a stored target representation. We compared the model's behavior at different mixtures of TD and BU components to the eye movement behavior of human observers performing the identical search task. We found that a purely TD model provides a much closer match to human behavior than any mixture model using BU information. Only when biological constraints are removed (e.g., eliminating the retina) did a BU/TD mixture model begin to approximate human behavior.

## 1. Introduction

The human object detection literature, also known as visual search, has long struggled with how best to conceptualize the role of bottom-up (BU) and top-down (TD) processes in guiding search behavior.[1] Early theories of search assumed a pure BU feature decomposition of the objects in an image, followed by the later reconstitution of these features into objects if the object's location was visited by spatially directed visual attention [1]. Importantly, the direction of attention to feature locations was believed to be random in these early models, thereby making them devoid of any BU or TD component contributing to the guidance of attention to objects in scenes.

The belief in a random direction of attention during search was quashed by Wolfe and colleague's [2] demonstration of TD information affecting search guidance. According to their guided-search model [3], preattentively available features from objects not yet bound by attention can be compared to a high-level target description to generate signals indicating evidence for the target in a display. The search process can then use these signals to

guide attention to display locations indicating the greatest evidence for the target. More recent models of TD target guidance can accept images of real-world scenes as stimuli and generate sequences of eye movements that can be directly compared to human search behavior [4].

Purely BU models of attention guidance have also enjoyed a great deal of recent research interest. Building on the concept of a saliency map introduced in [5], these models attempt to use biologically plausible computational primitives (e.g., center-surround receptive fields, color opponency, winner-take-all spatial competition, etc.) to define points of high salience in an image that might serve as attractors of attention. Much of this work has been discussed in the context of scene perception [6], but recently Itti and Koch [7] extended a purely BU model to the task of visual search. They defined image saliency in terms of intensity, color, and orientation contrast for multiple spatial scales within a pyramid. They found that a saliency model based on feature-contrast was able to account for a key finding in the behavioral search literature, namely very efficient search for feature-defined targets and far less efficient search for targets defined by conjunctions of features [1].

Given the body of evidence suggesting both TD and BU contributions to the guidance of attention in a search task, the logical next question to ask is whether these two sources of information should be combined to describe search behavior and, if so, in what proportion? To answer this question, we adopt a three-pronged approach. First, we implement two models of eye movements during visual search, one a TD model derived from the framework proposed by [4] and the other a BU model based on the framework proposed by [7]. Second, we use an eyetracker to collect behavioral data from human observers so as to quantify guidance in terms of the number of fixations needed to acquire a target. Third, we combine the outputs of the two models in various proportions to determine the TD/BU weighting best able to describe the number of search fixations generated by the human observers.

## 2. Eye movement model

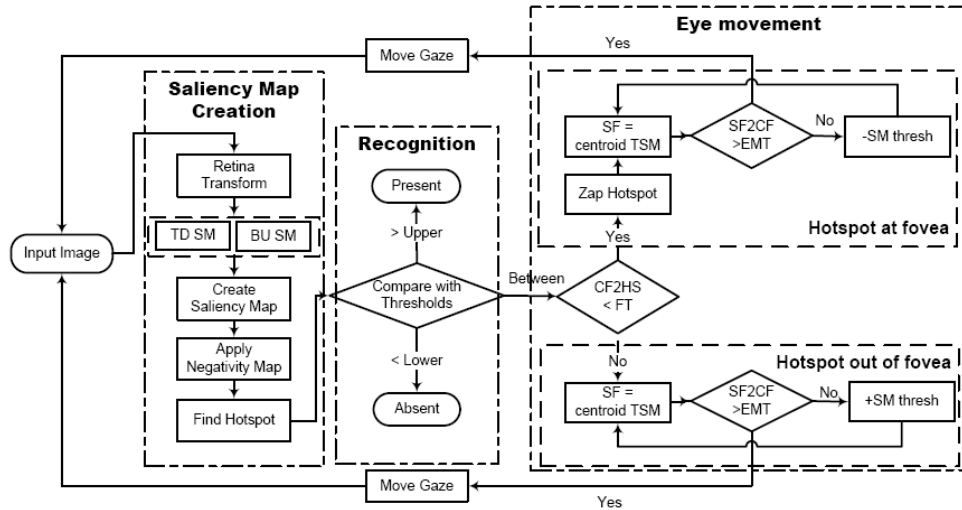

Figure 1: Flow of processing through the model. Abbreviations: TD SM (top-down saliency map); BU SM (bottom-up saliency map); SF(suggested fixation point); TSM (thresholded saliency map); CF2HS (Euclidean distance between current fixation and hotspot); SF2CF(Euclidean distance between suggested fixation and current fixation); EMT (eye movement threshold); FT (foveal threshold).

In this section we introduce a computational model of eye movements during visual search. The basic flow of processing in this model is shown in Figure 1. Generally, we repre-

sent search scenes in terms of simple and biologically-plausible visual feature-detector responses (colors, orientations, scales). Visual routines then act on these representations to produce a sequence of simulated eye movements. Our framework builds on work described in [8, 4], but differs from this earlier model in several important respects. First, our model includes a perceptually-accurate simulated retina, which was not included in [8, 4]. Second, the visual routine responsible for moving gaze in our model is fundamentally different from the earlier version. In [8, 4], the number of eye movements was largely determined by the number of spatial scale filters used in the representation. The method used in the current model to generate eye movements (Section 2.3) removes this upper limit. Third, and most important to the topic of this paper, the current model is capable of integrating both BU and TD information in guiding search behavior. The [8, 4] model was purely TD.

## 2.1. Overview

The model can be conceptually divided into three broad stages: (1) the creation of a saliency map (SM) based on TD and BU analysis of a retinally-transformed image, (2) recognizing the target, and (3) the operations required to generate eye movements. Within each of these stages are several more specific operations, which we will now describe briefly in an order determined by the processing flow.

**Input image**: The model accepts as input a high-resolution ($1280 \times 960$ pixel) image of the search scene, as well as a smaller image of the search target. A point is specified on the target image and filter responses are collected from a region surrounding this point. In the current study this point corresponded to the center of the target image.

**Retina transform**: The search image is immediately transformed to reflect the acuity limitations imposed by the human retina. To implement this neuroanatomical constraint, we adopt a method described in [9], which was shown to provide a good fit to acuity limitations in the human visual system. The approach takes an image and a fixation point as input, and outputs a retina-transformed version of the image based on the fixation point (making it a good front-end to our model). The initial retina transformation assumes fixation at the center of the image, consistent with the behavioral experiment. A new retina transformation of the search image is conducted after each change in gaze.

**Saliency maps**: Both the TD and the BU saliency maps are based on feature responses from Gaussian filters of different orientations, scales, colors, and orders. These two maps are then combined to create the final SM used to guide search (see Section 2.2 for details).

**Negativity map**: The negativity map keeps a spatial record of every nontarget location that was fixated and rejected through the application of Gaussian inhibition, a process similar to *inhibition of return* [10] that we refer to as "zapping". The existence of such a map is supported by behavioral evidence indicating a high-capacity spatial memory for rejected nontargets in a search task [11].

**Find hotspot**: The hotspot (HS) is defined as the point on the saliency map having the largest saliency value. Although no biologically plausible mechanism for isolating the hotspot is currently used, we assume that a standard winner-take-all (WTA) algorithm can be used to find the SM hotspot.

**Recognition thresholds**: Recognition is accomplished by comparing the hotspot value with two thresholds. The model terminates with a target-present judgment if the hotspot value exceeds a high target-present threshold, set at .995 in the current study. A target-absent response is made if the hotspot value falls below a low target-absent threshold (not used in the current study). If neither of these termination criteria are satisfied, processing passes to the eye movement stage.

**Foveal threshold**: Processing in the eye movement stage depends on whether the model's simulated fovea is fixated on the SM hotspot. This event is determined by computing the Euclidean distance between the current location of the fovea's center and the hotspot (CF2HS), then comparing this distance to a foveal threshold (FT). The FT, set at 0.5 deg

of visual angle, is determined by the retina transform and viewing angle and corresponds to the radius of the foveal window size. The foveal window is the region of the image not blurred by the retina transform function, much like the high-resolution foveola in the human visual system.

**Hotspot out of fovea**: If the hotspot is not within the FT, meaning that the object giving rise to the hotspot is not currently fixated, then the model will make an eye movement to bring the simulated fovea closer to the hotspot's location. In making this movement, the model will be effectively canceling the effect of the retina transform, thereby enabling a judgment regarding the hotspot pattern. The destination of the eye movement is computed by taking the weighted centroid of activity on the thresholded saliency map (TSM). See Section 2.3 for additional details regarding the centroid calculation of the suggested fixation point (SF), its relationship to the distance threshold for generating an eye movement (EMT), and the dynamically-changing threshold used to remove those SM points offering the least evidence for the target (+SM thresh).

**Hotspot at fovea**: If the simulated fovea reaches the hotspot (CF2HS < FT) and the target is still not detected (HS < target-present threshold), the model is likely to have fixated a nontarget. When this happens (a common occurrence in the course of a search), it is desirable to inhibit the location of this false target so as not to have it re-attract attention or gaze. To accomplish this, we inhibit or "zap" the hotspot by applying a negative Gaussian filter centered at the hotspot location (set at $63$ pixels). Following this injection of negativity into the SM, a new eye movement is made based on the dynamics outlined in Section 2.3.

## 2.2. Saliency map creation

The first step in creating the TD and BU saliency maps is to separate the retina-transformed image into an intensity channel and two opponent-process color channels (R-G and B-Y). For each channel, we then extract visual features by applying a set of steerable 2D Gaussian-derivative filters, $G(t, \theta, s)$, where $t$ is the order of the Gaussian kernel, $\theta$ is the orientation, and $s$ is the spatial scale. The current model uses first and second order Gaussians, 4 orientations (0, 45, 90 and 180 degrees), and 3 scales (7, 15 and 31 pixels), for a total of 24 filters. We therefore obtain 24 feature maps of filter responses per channel, $M(t, \theta, s)$, or alternatively, a 72-dimensional feature vector, $F$, for each pixel in the retina-transformed image.

The TD saliency map is created by correlating the retina-transformed search image with the target feature vector $F_t$.[2]

To maintain consistency between the two saliency map representations, the same channels and features used in the TD saliency map were also used to create the BU saliency map. Feature-contrast signals on this map were obtained directly from the responses of the Gaussian derivative filters. For each channel, the 24 feature maps were combined into a single map according to:

$$\sum_{t,\theta,s} \mathcal{N}(|M(t,\theta,s)|) \tag{1}$$

where $\mathcal{N}(\bullet)$ is the normalization function described in [12]. The final BU saliency map is then created by averaging the three combined feature maps. Note that this method of creating a BU saliency map differs from the approach used in [12, 7] in that our filters consisted of $1^{st}$ and $2^{nd}$ order derivatives of Gaussians and not center-surround DoG filters. While the two methods of computing feature contrast are not equivalent, in practice they yield very similar patterns of BU salience.

Finally, the combined SM was simply a linear combination of the TD and BU saliency maps, where the weighting coefficient was a parameter manipulated in our experiments.

### 2.3. Eye movement generation

Our model defines gaze position at each moment in time by the weighted spatial average (centroid) of signals on the SM, a form of neuronal population code for the generation of eye movement [13, 14]. Although a centroid computation will tend to bias gaze in the direction of the target (assuming that the target is the maximally salient pattern in the image), gaze will also be pulled away from the target by salient nontarget points. When the number of nontarget points is large, the eye will tend to move toward the geometric center of the scene (a tendency referred to in the behavioral literature as the global effect, [15, 16]); when the number of points is small, the eye will move more directly to the target.

To capture this activity-dependent eye movement behavior, we introduce a moving threshold, $\rho$, that excludes points from the SM over time based on their signal strength. Initially $\rho$ will be set to zero, allowing every signal on the SM to contribute to the centroid gaze computation. However, with each timestep, $\rho$ is increased by .001, resulting in the exclusion of minimally salient points from the SM (+ SM thresh in Figure 1). The centroid of the SM, what we refer to as the suggested fixation point (SF), is therefore dependent on the current value of $\rho$ and can be expressed as:

$$SF = \sum_{S_p > \rho} \frac{pS_p}{\sum S_p}.$$ (2)

Eventually, only the most salient points will remain on the thresholded saliency map (TSM), resulting in the direction of gaze to the hotspot. If this hotspot is not the target, $\rho$ can be decreased (- SM thresh in Figure 1) after zapping in order to reintroduce points to the SM. Such a moving threshold is a plausible mechanism of neural computation easily instantiated by a simple recurrent network [17].

In order to prevent gaze from moving with each change in $\rho$, which would result in an unrealistically large number of very small eye movements, we impose an eye movement threshold (EMT) that prevents gaze from shifting until a minimum distance between SF and CF is achieved (SF2CF > EMT in Figure 1). The EMT is based on the signal and noise characteristics of each retina-transformed image, and is defined as:

$$EMT = \max\left(FT, d(1 + Cd \log \frac{Signal}{Noise})\right),$$ (3)

where $FT$ is the fovea threshold, $C$ is a constant, and $d$ is the distance between the current fixation and the hotspot. The $Signal$ term is defined as the sum of all foveal saliency values on the TSM; the $Noise$ term is defined as the sum of all other TSM values. The *Signal/Noise log ratio* is clamped to the range of $[-1/C, 0]$. The lower bound of the SF2CF distance is $FT$, and the upper bound is $d$. The eye movement dynamics can therefore be summarized as follows: incrementing $\rho$ will tend to increase the SF2CF distance, which will result in an eye movement to SF once this distance exceeds the EMT.

## 3. Experimental methods

For each trial, the two human observers and the model were first shown an image of a target (a tank). In the case of the human observers, the target was presented for one second and presumably encoded into working memory. In the case of the model, the target was represented by a single 72-dimensional feature vector as described in Section 2. A search image was then presented, which remained visible to the human observers until they made a button press response. Eye movements were recorded during this interval using an ELII eyetracker. Section 2 details the processing stages used by the model. There were 44 images and targets, which were all modified versions of images in the TNO dataset [18]. The images subtended approximately $20°$ on both the human and simulated retinas.

# 4. Experimental results

Model and human data are reported from 2 experiments. For each experiment we tested 5 weightings of TD and BU components in the combined SM. Expressed as a proportion of the BU component, these weightings were: BU 0 (TD only), BU .25, BU .5, BU .75, and BU 1.0 (BU only).

## 4.1. Experiment 1

Table 1: Human and model search behavior at 5 TD/BU mixtures in Experiment 1.

| *Retina* | Human subjects | | Model | | | | |
|---|---|---|---|---|---|---|---|
| *Population* | H1 | H2 | TD only | BU: 0.25 | BU: 0.5 | BU: 0.75 | BU only |
| Misses (%) | 0.00 | 0.00 | **0.00** | 36.36 | 72.73 | 77.27 | 88.64 |
| Fixations | 4.55 | 4.43 | **4.55** | 18.89 | 20.08 | 21.00 | 22.40 |
| Std Dev | 0.88 | 2.15 | **0.82** | 10.44 | 12.50 | 10.29 | 12.58 |

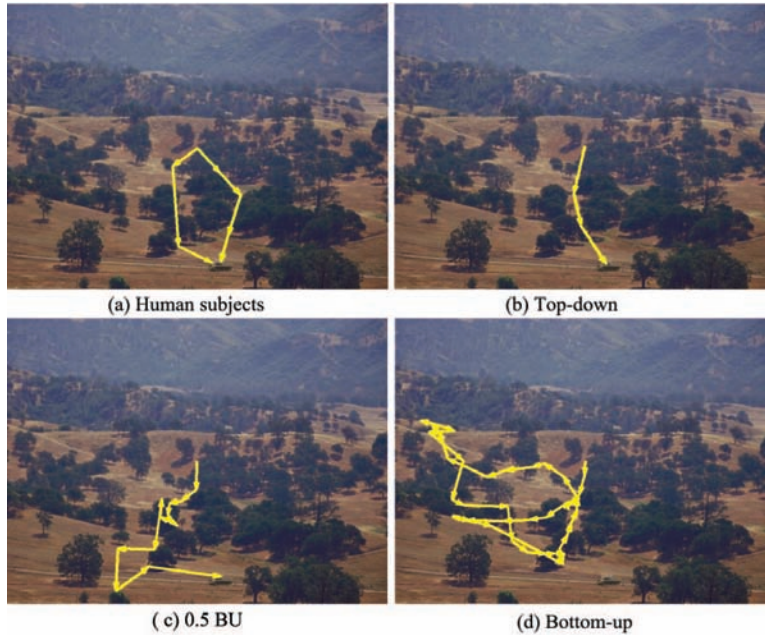

(a) Human subjects   (b) Top-down

( c) 0.5 BU   (d) Bottom-up

Figure 2: Comparison of human and model scanpaths at different TD/BU weightings.

As can be seen from Table 1, the human observers were remarkably consistent in their behavior. Each required an average of $4.5$ fixations to find the target (defined as gaze falling within .5 deg of the target's center), and neither generated an error (defined by a failure to find the target within $40$ fixations). Human target detection performance was matched almost exactly by a pure TD model, both in terms of errors ($0\%$) and fixations ($4.55$). This exceptional match between human and model disappeared with the addition of a BU component. Relative to the human and TD model, a BU $0.25$ mixture model resulted in a dramatic increase in the miss rate ($36\%$) and in the average number of fixations needed to acquire the target ($18.9$) on those trials in which the target was ultimately fixated. These high miss and fixation rates continued to increase with larger weightings of the BU contribution, reaching an unrealistic $89\%$ misses and 22 fixations with a pure BU model.

Figure 2 shows representative eye movement scanpaths from our two human observers (a) and the model at three different TD/BU mixtures (b, BU 0; c, BU 0.5; d, BU 1.0) for one image. Note the close agreement between the human scanpaths and the behavior of the

TD model. Note also that, with the addition of a BU component, the model's eye either wanders to high-contrast patterns (bushes, trees) before landing on the target (c), or misses the target entirely (d).

## 4.2. Experiment 2

Recently, Navalpakkam & Itti [19] reported data from a saliency-based model also integrating BU and TD information to guide search. Among their many results, they compared their model to the purely TD model described in [4] and found that their mixture model offered a more realistic account of human behavior. Specifically, they observed that the [4] model was too accurate, often predicting that the target would be fixated after only a single eye movement. Although our current findings would seem to contradict [19]'s result, this is not the case. Recall from Section 2.0 that our model differs from [4] in two respects: (1) it retinally transforms the input image with each fixation, and (2) it uses a thresholded population-averaging code to generate eye movements. Both of these additions would be expected to increase the number of fixations made by the current model relative to the TD model described in [4]. Adding a simulated retina should increase the number of fixations by reducing the target-scene TD correlations and increasing the probability of false targets emerging in the blurred periphery. Adding population averaging should increase fixations by causing eye movements to locations other than hotspots. It may therefore be the case that [19]'s critique of [4] may be pointing out two specific weaknesses of [4]'s model rather than a general weakness of their TD approach.

To test this hypothesis, we disabled the artificial retina and the population averaging code in our current model. The model now moves directly from hotspot to hotspot, zapping each before moving to the next. Without retinal blurring and population averaging, the behavior of this simpler model is now driven entirely by a WTA computation on the combined SM. Moreover, with a BU weighting of 1.0, this version of our model now more closely approximates other purely BU models in the literature that also lack retinal acuity limitations and population dynamics.

Table 2: Human and model search behavior at 5 TD/BU mixtures in Experiment 2.

| NO Retina | Human subjects | | Model | | | | |
|-----------|------|------|---------|----------|---------|----------|---------|
| NO Population | H1 | H2 | TD only | BU: 0.25 | BU: 0.5 | BU: 0.75 | BU only |
| Misses (%) | 0.00 | 0.00 | 0.00 | 9.09 | 27.27 | 56.82 | 68.18 |
| Fixations | 4.55 | 4.43 | 1.00 | 8.73 | 16.60 | 13.37 | 14.71 |
| Std Dev | 0.88 | 2.15 | 0.00 | 9.15 | 12.29 | 9.20 | 12.84 |

Table 2 shows the data from this experiment. The first two columns replot the human data from Table 1. Consistent with [19], we now find that the performance of a purely TD model is too good. The target is consistently fixated after only a single eye movement, unlike the $4.5$ fixations averaged by human observers. Also consistent with [19] is the observation that a BU contribution may assist this model in better characterizing human behavior. Although a $0.25$ BU weighting resulted in a doubling of the human fixation rate and $9\%$ misses, it is conceivable that a smaller BU weighting could nicely describe human performance. As in Experiment 1, at larger BU weightings the model again generated unrealistically high error and fixation rates. These results suggest that, in the absence of retinal and neuronal population-averaging constraints, BU information may play a small role in guiding search.

## 5. Conclusions

To what extent is TD and BU information used to guide search behavior? The findings from Experiment 1 offer a clear answer to this question: when biologically plausible constraints are considered, any addition of BU information to a purely TD model will worsen, not improve, the match to human search performance (see [20] for a similar conclusion applied to a walking task). The findings from Experiment 2 are more open to interpretation. It may be possible to devise a TD model in which adding a BU component might prove useful, but doing this would require building into this model biologically implausible assumptions.

A corollary to this conclusion is that, when these same biological constraints are added to existing BU saliency-based models, these models may no longer be able to describe human behavior.

A final fortuitous finding from this study is the surprising degree of agreement between our purely TD model and human performance. The fact that this agreement was obtained by direct comparison to human behavior (rather than patterns reported in the behavioral literature), and observed in eye movement variables, lends validity to our method. Future work will explore the generality of our TD model, extending it to other forms of TD guidance (e.g., scene context) and tasks in which a target may be poorly defined (e.g., categorical search).

**Acknowledgments**

This work was supported by a grant from the ARO (DAAD19-03-1-0039) to G.J.Z.

## Footnotes

[1]In this paper we will refer to BU guidance as guidance based on task-independent signals arising from basic neuronal feature analysis. TD guidance will refer to guidance based on information not existing in the input image or proximal search stimulus, such as knowledge of target features or processing constraints imposed by task instruction.

[2]Note that because our TD saliency maps are derived from correlations between target and scene images, the visual statistics of these images are in some sense preserved and might be described as a BU component in our model. Nevertheless, the correlation-based guidance signal requires knowledge of a target (unlike a true BU model), and for this reason we will continue to refer to this as a TD process.

# References

[1] A. Treisman and G. Gelade. A feature-integration theory of attention. *Cognitive Psychology*, 12:97–136, 1980.

[2] J. Wolfe, K. Cave, and S. Franzel. Guided search: An alternative to the feature integration model for visual search. *Journal of Experimental Psychology: Human Perception and Performance*, 15:419–433, 1989.

[3] J. Wolfe. Guided search 2.0: A revised model of visual search. *Psychonomic Bulletin and Review*, 1:202–238, 1994.

[4] R. Rao, G. Zelinsky, M. Hayhoe, and D. Ballard. Eye movements in iconic visual search. *Vision Research*, 42:1447–1463, 2002.

[5] C. Koch and S. Ullman. Shifts of selective visual attention: Toward the underlying neural circuitry. *Human Neurobiology*, 4:219–227, 1985.

[6] L. Itti and C. Koch. Computational modeling of visual attention. *Nature Reviews Neuroscience*, 2(3):194–203, 2001.

[7] L. Itti and C. Koch. A saliency-based search mechanism for overt and covert shift of visual attention. *Vision Research*, 40(10-12):1489–1506, 2000.

[8] R. Rao, G. Zelinsky, M. Hayhoe, and D. Ballard. Modeling saccadic targeting in visual search. In *NIPS*, 1995.

[9] J.S. Perry and W.S. Geisler. Gaze-contingent real-time simulation of arbitrary visual fields. In *SPIE*, 2002.

[10] R. M. Klein and W.J. MacInnes. Inhibition of return is a foraging facilitator in visual search. *Psychological Science*, 10(4):346–352, 1999.

[11] C. A. Dickinson and G. Zelinsky. Marking rejected distractors: A gaze-contingent technique for measuring memory during search. *Psychonomic Bulletin and Review*, In press.

[12] L. Itti, C. Koch, and E. Niebur. A model of saliency-based visual attention for rapid scene analysis. *PAMI*, 20(11):1254–1259, 1998.

[13] T. Sejnowski. Neural populations revealed. *Nature*, 332:308, 1988.

[14] C. Lee, W. Rohrer, and D. Sparks. Population coding of saccadic eye movements by neurons in the superior colliculus. *Nature*, 332:357–360, 1988.

[15] J. Findlay. Global visual processing for saccadic eye movements. *Vision Research*, 22:1033–1045, 1982.

[16] G. Zelinsky, R. Rao, M. Hayhoe, and D. Ballard. Eye movements reveal the spatio-temporal dynamics of visual search. *Psychological Science*, 8:448–453, 1997.

[17] J. L. Elman. Finding structures in time. *Cognitive Science*, 14:179–211, 1990.

[18] A. Toet, P. Bijl, F. L. Kooi, and J. M. Valeton. A high-resolution image dataset for testing search and detection models. Technical Report TNO-NM-98-A020, TNO Human Factors Research Institute,, Soesterberg, The Netherlands, 1998.

[19] V. Navalpakkam and L Itti. Modeling the influence of task on attention. *Vision Research*, 45:205–231, 2005.

[20] K. A. Turano, D. R. Geruschat, and F. H. Baker. Oculomotor strategies for direction of gaze tested with a real-world activity. *Vision Research*, 43(3):333–346, 2003.
